# Perceptual Multistability as Markov Chain Monte Carlo Inference

**Samuel J. Gershman**
Department of Psychology and Neuroscience Institute
Princeton University
Princeton, NJ 08540
`sjgershm@princeton.edu`

**Edward Vul & Joshua B. Tenenbaum**
Department of Brain and Cognitive Sciences
Massachusetts Institute of Technology
Cambridge, MA 02139
`{evul,jbt}@mit.edu`

## Abstract

While many perceptual and cognitive phenomena are well described in terms of Bayesian inference, the necessary computations are intractable at the scale of real-world tasks, and it remains unclear how the human mind approximates Bayesian computations algorithmically. We explore the proposal that for some tasks, humans use a form of Markov Chain Monte Carlo to approximate the posterior distribution over hidden variables. As a case study, we show how several phenomena of perceptual multistability can be explained as MCMC inference in simple graphical models for low-level vision.

## 1   Introduction

People appear to make rational statistical inferences from noisy, uncertain input in a wide variety of perceptual and cognitive domains [1, 9]. However, the computations for such inference, even for relatively small problems, are often intractable. For larger problems like those people face in the real world, the space of hypotheses that must be entertained is infinite. So how can people achieve solutions that seem close to the Bayesian ideal? Recent work has suggested that people may use approximate inference algorithms similar to those used for solving large-scale problems in Bayesian AI and machine learning [23, 4, 14]. "Rational models" of human cognition at the level of computational theories are often inspired by models for analogous inferences in machine learning. In the same spirit of reverse engineering cognition, we can also look to the general-purpose approximation methods used in these engineering fields as the inspiration for "rational process models"—principled algorithmic models for how Bayesian computations are implemented approximately in the human mind.

Several authors have recently proposed that humans approximate complex probabilistic inferences by sampling [19, 14, 21, 6, 4, 24, 23], constructing Monte Carlo estimates similar to those used in Bayesian statistics and AI [16]. A variety of psychological phenomena have natural interpretations in terms of Monte Carlo methods, such as resource limitations [4], stochastic responding [6, 23] and order effects [21, 14]. The Monte Carlo methods that have received most attention to date as rational process models are importance sampling and particle filtering, which are traditionally seen as best suited to certain classes of inference problems: static low dimensional models and models with explicit sequential structure, respectively. Many problems in perception and cognition, however,

require inference in high dimensional models with sparse and noisy observations, where the correct global interpretation can only be achieved by propagating constraints from the ambiguous local information across the model. For these problems, *Markov Chain Monte Carlo* (MCMC) methods are often the method of choice in AI and machine vision [16]. Our goal in this paper is to explore the prospects for rational process models of perceptual inference based on MCMC.

MCMC refers to a family of algorithms that sample from the joint posterior distribution in a high-dimensional model by gradually drifting through the hypothesis space of complete interpretations, following a Markov chain that asymptotically spends time at each point in the hypothesis space proportional to its posterior probability. MCMC algorithms are quite flexible, suitable for a wide range of approximate inference problems that arise in cognition, but with a particularly long history of application in visual inference problems ([8] and many subsequent papers).

The chains of hypotheses generated by MCMC shows characteristic dynamics distinct from other sampling algorithms: the hypotheses will be temporally correlated and as the chain drifts through hypothesis space, it will tend to move from regions of low posterior probability to regions of high probability; hence hypotheses will tend to cluster around the modes. Here we show that the characteristic dynamics of MCMC inference in high-dimensional, sparsely coupled spatial models correspond to several well-known phenomena in visual perception, specifically the dynamics of multistable percepts.

Perceptual multistability [13] has long been of interest both phenomenologically and theoretically for models of perception as Bayesian inference [7, 20, 22, 10]. The classic example of perceptual multistability is the Necker cube, a 2D line drawing of a cube perceived to alternate between two different depth configurations (Figure 1A). Another classic phenomenon, extensively studied in psychophysics but less well known outside the field, is binocular rivalry [2]: when incompatible images are presented to the two eyes, subjects report a percept that alternates between the images presented to the left eye and that presented to the right (e.g., Figure 1B).

Bayesian modelers [7, 20, 22, 10] have interpreted these multistability phenomena as reflections of the shape of the posterior distribution arising from ambiguous observations, images that could have plausibly been generated by two or more distinct scenes. For the Necker cube, two plausible depth configurations have indistinguishable 2D projections; with binocular rivalry, two mutually exclusive visual inputs have equal perceptual fidelity. Under these conditions, the posterior over scene interpretations is bimodal, and rivalry is thought to reflect periodic switching between the modes. Exactly how this "mode-switching" relates to the mechanisms by which the brain implements Bayesian perceptual inference is less clear, however. Here we explore the hypothesis that the dynamics of multistability can be understood in terms of the output of an MCMC algorithm, drawing posterior samples in spatially structured probabilistic models for image interpretation.

Traditionally, bistability has been explained in non-rational mechanistic terms, for example, in terms of physiological mechanisms for adaptation or reciprocal inhibition between populations of neurons. Dayan [7] studied network models for Bayesian perceptual inference that estimate the *maximum a posteriori* scene interpretation, and proposed that multistability might occur in the presence of a multimodal posterior due to an additional neural oscillatory process whose function is specifically to induce mode-switching. He speculated that this mechanism might implement a form of MCMC inference but he did not pursue the connection formally. Our proposal is most closely related to the work of Sundareswara and Schrater [20, 22], who suggested that mode-switching in Necker cube-type images reflects a rational sampling-based algorithm for approximate Bayesian inference and decision making. They presented an elegant sampling scheme that could account for Necker cube bistability, with several key assumptions: (1) that the visual system draws a sequence of samples from the posterior over scene interpretations; (2) that the posterior probability of each sample is known; (3) that samples are weighted based on the product of their posterior probabilities and a memory decay process favoring more recently drawn samples; and (4) that perceptual decisions are made deterministically based on the sample with highest weight.

Our goal here is a simpler analysis that comes closer to the standard MCMC approaches used for approximate inference in Bayesian AI and machine vision, and establishing a clearer link between the mechanisms of perception in the brain and rational approximate inference algorithms on the engineering side. As in most applications of Bayesian inference in machine vision [8, 16], we do not assume that the visual system has access to the full posterior distribution over scene interpretations,

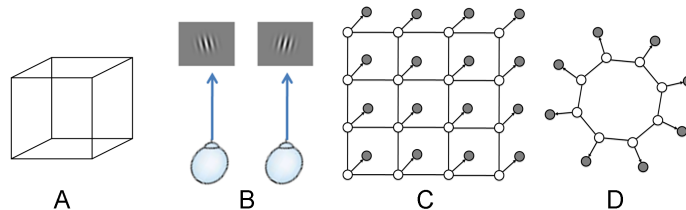

Figure 1: (*A*) Necker cube. (*B*) Binocular rivalry stimuli. (*C*) Markov random field image model with lattice and ring (*D*) topologies. Shaded nodes correspond to observed variables; unshaded nodes correspond to hidden variables.

which is expected to be extremely high-dimensional and complex. The visual system might be able to evaluate only relative probabilities of two similar hypotheses (as in Metropolis-Hastings), or to compute local conditional posteriors of one scene variable conditioned on its neighbors (as in Gibbs sampling). We also do not make extra assumptions about weighting samples based on memory decay, or require that conscious perceptual decisions be based on a memory for samples; consciousness has access to only the current state of the Markov chain, reflecting the observer's current brain state.

Here we show that several characteristic phenomena of multistability derive naturally from applying standard MCMC inference to Markov random fields (MRFs) – high dimensional, loosely coupled graphical models with spatial structure characteristic of many low-level and mid-level vision problems. Specifically, we capture the classic findings of Gamma-distributed mode-switching times in bistable perception; the biasing effects of contextual stimuli; the situations in which fused (rather than bistable) percepts occur, and the propagation of perceptual switches in traveling waves across the visual field. Although it is unlikely that this MCMC scheme corresponds *exactly* to any process in the visual system, and it is almost surely too simplified or limited as a general account of perceptual multistability, our results suggest that MCMC could provide a promising foundation on which to build rational process-level accounts of human perception and perhaps cognition more generally.

## 2   Markov random field image model

Our starting point is a simple and schematic model of vision problems embodying the idea that images are generated by a set of hidden variables with local dependencies. Specifically, we assume that each observed image element $x_i$ is connected to a hidden variable $z_i$ by a directed edge, and each hidden variable is connected to its neighbors (in set $c_i$) by an undirected edge (thus implying that each hidden variable is conditionally independent of all others given its neighbors). This Markov property is often exploited in computer vision [8] because elements of an image tend to depend on their adjacent neighbors, but are less influenced by more distant elements. Formally, this assumption corresponds to a *Markov random field* (MRF). Different topologies of the MRF (e.g., lattice or ring) can be used to capture the structure of different visual objects (Figure 1C,D). The joint distribution over configurations of hidden and observed variables is given by:

$$P(\mathbf{z}, \mathbf{x}) = Z^{-1} \exp \left[ - \sum_i R(x_i|z_i) - V(z_i|\mathbf{z}_{c_i}) \right], \tag{1}$$

where $Z$ is a normalizing constant, and $R$ and $V$ are *potential functions*. In a Gaussian MRF, the conditional potential function over hidden node $i$ is given by

$$V(z_i|\mathbf{z}_{c_i}) = \mu_i - \lambda \sum_{j \in c_i} (z_i - z_j)^2, \tag{2}$$

where $\lambda$ is a precision (inverse variance) parameter specifying the coupling between neighboring hidden nodes; when $\lambda$ is large, a node will be strongly influenced by its neighbors. The $\mu_i$ term represents the prior mean of $z_i$, which can be used to encode contextual biases, as we discuss below.

We construct the likelihood potential $R(x_i|z_i)$ to express the ambiguity of the image by making it multimodal: several different hidden causes are equally likely to have generated the image. Since

for our purposes only the likelihood of $x_i$ matters, we can arbitrarily set $x_i = 0$ and formalize the multimodal likelihood as a mixture of Gaussians evaluated at points $a$ and $b$:

$$R(x_i|z_i) = \mathcal{N}(z_i; a, \sigma^2) + \mathcal{N}(z_i; b, \sigma^2). \tag{3}$$

The computational problem for vision (as we are framing it) is to infer the hidden causes of an observed image. Given an observed image $\mathbf{x}$, the posterior distribution over hidden causes $\mathbf{z}$ is

$$P(\mathbf{z}|\mathbf{x}) = \frac{P(\mathbf{x}|\mathbf{z})P(\mathbf{z})}{\int_{\mathbf{z}} P(\mathbf{x}|\mathbf{z})P(\mathbf{z})d\mathbf{z}}. \tag{4}$$

There are a number of reasons why Equation 4 may be computationally intractable. One is that the integration in the denominator may be high dimensional and lacking an analytical solution. Another is that there may not exist a simple functional form for the posterior. Assuming it is intractable to perform exact inference, we now turn to approximate solutions based on sampling.

## 3   Markov chain monte carlo

The basic idea behind Monte Carlo methods is to approximate a distribution with a set of samples drawn from that distribution. In order to use Monte Carlo approximations, one must be able to draw samples from the posterior, but it is often impossible to do so directly. MCMC methods address this problem by drawing samples from a Markov chain that converges to the posterior distribution [16]. There are many variations of MCMC methods but here we will focus on the simplest: the Metropolis algorithm [18]. Each step of the algorithm consists of two stages: a *proposal* stage and an *acceptance* stage. An accepted proposal is a sample from a Markov chain that provably converges to the posterior. We will refer to $\mathbf{z}^{(l)}$ as the "state" at step $l$. In the proposal stage, a new state $\mathbf{z}'$ is proposed by generating a random sample from a proposal density $Q\left(\mathbf{z}'; \mathbf{z}^{(l)}\right)$ that depends on the current state. In the acceptance stage, this proposal is accepted with probability

$$P\left(\mathbf{z}^{(l+1)} = \mathbf{z}'|\mathbf{z}^{(l)}\right) = \min\left[1, \frac{P(\mathbf{z}'|\mathbf{x})}{P\left(\mathbf{z}^{(l)}|\mathbf{x}\right)}\right], \tag{5}$$

where we have assumed for simplicity that the proposal is symmetric: $Q(\mathbf{z}'; \mathbf{z}) = Q(\mathbf{z}; \mathbf{z}')$. If the proposal is rejected, the current state is repeated in the chain.

## 4   Results

We now show how the Metropolis algorithm applied to the MRF image model gives rise to a number of phenomena in binocular rivalry experiments. Unless mentioned otherwise, we use the following parameters in our simulations: $\mu = 0, \lambda = 0.25, \sigma = 0.1, a = 1, b = -1$. For the ring topology, we used $\lambda = 0.2$ to compensate for the fewer neighbors around each node as compared to the lattice topology. The sampler was run for $200,000$ iterations. For some simulations, we systematically manipulated certain parameters to demonstrate their role in the model. We have found that the precise values of these parameters have relatively little effect on the model's behavior. For all simulations we used a Gaussian proposal (with standard deviation 1.5) that alters the state of one hidden node (selected at random) on each iteration.

### 4.1   Distribution of dominance durations

One of the most robust findings in the literature on perceptual multistability is that switching times in binocular rivalry between different stable percepts tend to follow a Gamma-like distribution. In other words, the "dominance" durations of stability in one mode tend to be neither overwhelmingly short nor long. This effect is so characteristic of binocular rivalry that there have been countless psychophysical experiments measuring the differences in Gamma switching time parameters across manipulations, and testing whether Gamma, or log-normal distributions are best [2]. To account for this characteristic behavior, many papers have described neural circuits that could produce switching oscillations with the right stochastic dynamics (e.g., [25]). Existing rational process models of multistability [7, 20, 22] likewise appeal to specific implementational-level constraints to produce

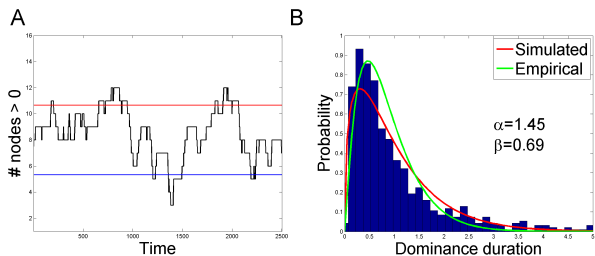

Figure 2: (*A*) Simulated timecourse of bistability in the lattice MRF. Plotted on the y-axis is the number of nodes with value greater than 0. The horizontal lines show the thresholds for a perceptual switch. (*B*) Distribution of simulated dominance durations (mean-normalized) for MRF with lattice topology. Curves show gamma distributions fitted to simulated (with parameter values shown on the right) and empirical data, replotted from [17]

this effect. In contrast, here we show how Gamma-distributed dominance durations fall naturally out of MCMC operating on an MRF.

We constructed a $4 \times 4$ grid to model a typical binocular rivalry grating. In the typical experiment reporting a Gamma distribution of dominance durations, subjects are asked to say which of two images corresponds to their "global" percept. To make the same query of the current state of our simulated MCMC chain, we defined a perceptual switch to occur when at least $2/3$ of the hidden nodes turn positive or negative. Figure 2A shows a sample of the timecourse[1] and the distribution of dominance durations and maximum-likelihood estimates for the Gamma parameters $\alpha$ (shape) and $\beta$ (scale), demonstrating that the durations produced by MCMC are well-described by a Gamma distribution (Figure 2B).

It is interesting to note that the MRF structure of the problem (representing the multivariate structure of low-level vision) is an important pre-condition to obtaining a Gamma-like distribution of dominance durations: When considering MCMC on only a single node, the measured dominance durations tend to be exponentially-distributed. The Gamma distribution may arise in MCMC on an MRF because each hidden node takes an exponentially-distributed amount of time to switch (and these switches follow roughly one after another). In these settings, the total amount of time until enough nodes switch to one mode will be Gamma-distributed (i.e., the sum of exponentially-distributed random variables is Gamma-distributed). [20, 22] also used this idea to explain mode-switching. In their model, each sample is paired with a weight initialized to the sample's posterior probability, and the sample with the largest weight designated as the dominant percept. Since multiple samples may correspond to the same percept, a particular percept will lose dominance only when the weights on all such samples decrease below the weights on samples of the non-dominant percept. By assuming an exponential decay on the weights, the time it takes for a single sample to lose dominance will be approximately exponentially distributed, leading to a Gamma distribution on the time it takes for multiple samples of the same percept to lose dominance. Here we have attempted to capture this effect within a rational inference procedure by attributing the exponential dynamics to the operation of MCMC on individual nodes in the MRF, rather than a memory decay process on individual samples.

## 4.2 Contextual biases

Much discussion in research on multistability revolves around the extent to which it is influenced by top-down processes like prior knowledge and attention [2]. In support of the existence of top-down

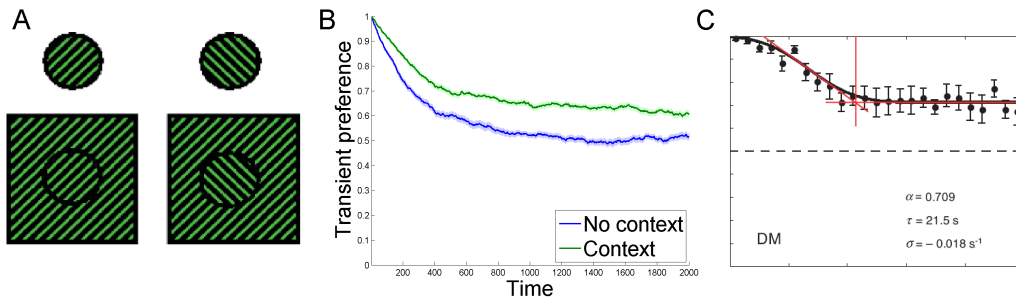

Figure 3: (*A*) Stimuli used by [5] in their experiment. On the top are the standard tilted grating patches presented dichoptically. On the bottom are the tilted grating patches superimposed on a background of rightward-tilting gratings, a contextual cue that biases dominance towards the rightward-tilting grating patch. (*B*) Simulated timecourse of transient preference for a lattice-topology MRF with and without a contextual cue (averaged over 100 runs of the sampler). (*C*) Empirical timecourse of transient preference fitted with a scaled cumulative Gaussian function, reprinted with permission from [17].

influences, several studies have shown that contextual cues can bias the relative dominance of rival stimuli. For example, [5] superimposed rivalrous tilted grating patches on a background of either rightward or leftward tilting gratings (Figure 3A) and showed that the direction of background tilt shifted dominance towards the monocular stimulus with context-compatible tilt. Following [20, 22], we modeled this result by assuming that the effect of context is to shift the prior mean towards the contextually-biased interpretation. We simulated this contextual bias by setting the prior mean $\mu = 1$. Figure 3B shows the timecourse of transient preference (probability of a particular interpretation at each timepoint) for the "context" and "no-context" simulations, illustrating this persistent bias.

Another property of this timeseries is the initial bias exhibited by both the context and no-context conditions, a phenomenon observed experimentally [17, 22] (Figure 3C). In fact, this is a distinctive property of Markov chains (as pointed out by [22]): MCMC algorithms generally take multiple iterations before they converge to the stationary distribution [16]. This initial period is known as the "burn-in." Thus, human perceptual inference may similarly require an initial burn-in period to reach the stationary distribution.

## 4.3 Deviations from stable rivalry: fusion

Most models have focused on the "stable" portions of the bistable dynamics of rivalry; however, in addition to the mode-hopping behavior that characterizes this phenomenon, bistable percepts often produce other states. In some conditions the two percepts are known to fuse, rather than rival: the percept then becomes a composite or superposition of the two stimuli (and hence no alternation is perceived). This fused perceptual state can be induced most reliably by decreasing the distance in feature space between the two stimuli [11] (Figure 4B) or decreasing the contrast of both stimuli [15]. These relations are shown schematically in Figure 4A. Neither neural, nor algorithmic, nor computational models of rivalry have thus far attempted to explain these findings.

In experiments on "fusion", subjects are given three options to report their percept: one of two global precepts or something in between. We define such a fused percept as a perceptual state lying between the two "bistable" modes — that is, an interpretation between the two rivalrous, high-probability interpretations. We can interpret manipulation of feature space distance in terms of the distance between the modes, and reductions of contrast as increases in the variance around the modes. When such manipulations are introduced to the MRF model, the posterior distribution changes as in Figure 4A (inset). By making the modes closer together or increasing the variance around the modes, greater probability mass is assigned to an intermediate zone between the modes—a fused percept. Thus, manipulating stimulus separation (feature distance) or stimulus fidelity (contrast) changes the parameterizations of the likelihood function, and these manipulations produce systematically increasing odds of fused percepts, matching the phenomenology of these stimuli (Figure 4B).

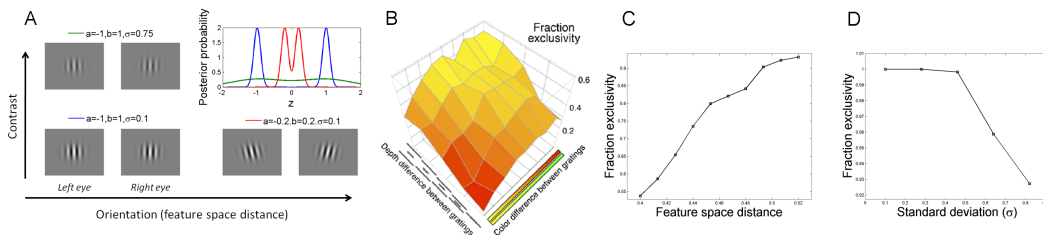

Figure 4: (*A*) Schematic illustration of manipulating orientation (feature space distance) and contrast in binocular rivalry stimuli. The inset shows effects of different likelihood parameterizations on the posterior distribution, designed to mimic these experimental manipulations. (*B*) Experimental effects of increasing feature space distance (depth and color difference) between rivalrous gratings on exclusivity of monocular percepts, reprinted with permission from [11]. Increasing the distance in feature space between rivalrous stimuli (*C*) or the contrast of both stimuli (*D*), modeled as increasing the variance around the modes, increases the probability of observing an exclusive percept in simulations.

## 4.4 Traveling waves

Fused percepts are not the only deviations from bistability. In other circumstances, particularly in binocular rivalry, stability is often incomplete across the visual field, producing "piecemeal" rivalry, in which one portion of the visual field looks like the image in one eye, while another portion looks like the image in the other eye. One tantalizing feature of these piecemeal percepts is the phenomenon known as traveling waves: subjects tend to perceive a perceptual switch as a "wave" propagating over the visual field [26, 12]: the suppressed stimulus becomes dominant in an isolated location of the visual field and then gradually spreads. These traveling waves reveal an interesting local dynamics during an individual switch itself, rather than just the Gamma-distributed dynamics of the time between complete switches of dominance. Like fused percepts, these intra-switch dynamics have been generally ignored by models of multistability.

Demonstrating the dynamics of traveling waves within patches of the percept requires a different method of probing perception. Wilson et al. [26] used annular stimuli (Figure 5A), and probed a particular patch along the annulus; they showed that the time at which the suppressed stimulus in the test patch becomes dominant is a function of the distance (around the circumference of the annulus) between the test patch and the patch where a dominance switch was induced by transiently increasing the contrast of the suppressed stimulus. This dependence of switch-time on distance (Figure 5B) suggested to Wilson et al. that stimulus dominance was propagating around the annulus. Using fMRI, Lee et al. [12] showed that the propagation of this "traveling wave" can be measured in primary visual cortex (V1; Figure 5): they used the retinotopic structure of V1 to identify brain regions corresponding to different portions of the the visual field, then measured the timing of the response in these regions to the induced dominance switch as a function of the cortical distance from the location of the initial switch. They found that the temporal delay in the response increased as a function of cortical distance from the V1 representation of the top of the annulus (Figure 5C).

To simulate such traveling waves within the percept of a stimulus, we constructed an MRF with ring topology and measured the propagation time (the time at which a mode-switch occurs) at different hidden nodes along the ring. To simulate the transient increase in contrast at one location to induce a switch, we initialized one node's state to be $+1$ and the rest to be $-1$. Consistent with the idea of wave propagation, Figure 5D shows the average time for a simulated node to switch modes as a function of distance around the ring. Intuitively, nodes will tend to switch in a kind of "domino effect" around the ring; the local dependencies in the MRF ensure that nodes will be more likely to switch modes once their neighbors have switched. Thus, once a switch at one node has been accepted by the Metropolis algorithm, a switch at its neighbor is likely to follow.

## 5 Conclusion

We have proposed a "rational process" model of perceptual multistability based on the idea that humans approximate the posterior distribution over the hidden causes of their visual input with a set of samples. In particular, the dynamics of the sample-generating process gives rise to much of

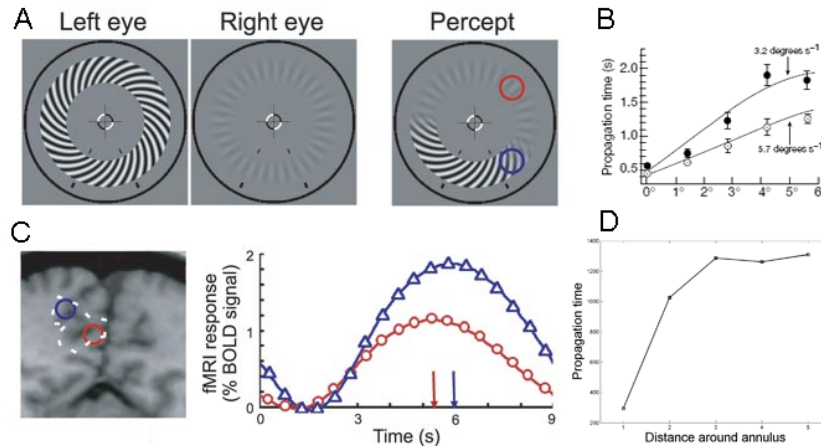

Figure 5: Traveling waves in binocular rivalry. (*A*) Annular stimuli used by Lee et al. (left and center panels) and the subject percept reported by observers (right panel), in which the low contrast stimulus was seen to spread around the annulus, starting at the top. Figure reprinted with permission from [12]. (*B*) Propagation time as a function of distance around the annulus, replotted from [26]. Filled circles represent radial gratings, open circles represent concentric gratings. (*C*) Anatomical image (left panel) showing the retinotopically-mapped coordinates of the initial and probe locations in V1. Right panel shows the measured fMRI responses for the two outlined subregions. (*D*) A transient increase in contrast of the suppressed stimulus induces a perceptual switch at the location of contrast change. The propagation time for a switch at a probe location increases with distance (around the annulus) from the switch origin.

the rich dynamics in multistable perception observed experimentally. These dynamics may be an approximation to the MCMC algorithms standardly used to solve difficult inference problems in machine learning and statistics [16].

The idea that perceptual multistability can be construed in terms of sampling in a Bayesian model was first proposed by [20, 22], and our work follows theirs closely in several respects. However, we depart from that work in the theoretical underpinnings of our model: It is not transparent how well the sampling scheme in [22, 24] approximates Bayesian inferences, or how it corresponds to standard algorithms where the full posterior is not assumed to be available when drawing samples. Our goal here is to show how some of the basic phenomena of multistable perception can be understood straightforwardly as the output of familiar, simple and effective methods for approximate inference in Bayesian machine vision.

A related point of divergence between our model and that of [20, 22], as well as other Bayesian models of multistable perception [7, 10], is that we are able to explain multistable perception in terms of a well-defined inference procedure that doesn't require ad-hoc appeals to neurophysiological processes like noise, adaptation, inhibition, etc. Thus, our contribution is to show how an inference algorithm widely used in statistics and computer science can give rise naturally to perceptual multistability phenomena. Of course, we do not wish to argue that neurophysiological processes are irrelevant. Our goal here was to abstract away from implementational details and make claims about the algorithmic level. Clearly an important avenue for future work is relating algorithms like MCMC to neural processes (indeed this connection was suggested previously by [7]).

Another important direction in which to extend this work is from rivalry with low-level stimuli to more complex vision problems that involve global coherence over the image (such as in natural scenes). Although similar perceptual dynamics have been observed with a wide range of ambiguous stimuli, the absence of obvious transition periods with the Necker cube suggests that these dynamics may differ in important ways from perception of rivalry stimuli.

**Acknowledgments:** This work was supported by ONR MURI: Complex Learning and Skill Transfer with Video Games N00014-07-1-0937 (PI: Daphne Bavelier); NDSEG fellowship to EV and NSF DRMS Dissertation grant to EV.

## Footnotes

[1]It may seem surprising that the model spends relatively little time near the extremes, and that switches are fairly gradual. This is not the phenomenology of bistability in a Necker cube, but it is the phenomenology of binocular rivlary with grating-like stimuli where experiments have shown that substantial time is spent in transition periods [3]. It seems that this is the case in scenarios where a simple planar MRF with nearest neighbor smoothness like the one we're considering is a good model. To capture the perception of depth in the Necker cube, or rivalry with more complex higher-level stimuli (like natural scenes), a more complex and densely interconnected graphical model would be required — in such cases the perceptual switching dynamics will be different.

# References

[1] J.R. Anderson. *The adaptive character of thought*. Lawrence Erlbaum Associates, 1990.

[2] R. Blake. A primer on binocular rivalry, including current controversies. *Brain and Mind*, 2(1):5–38, 2001.

[3] J.W. Brascamp, R. van Ee, A.J. Noest, RH Jacobs, and A.V. van den Berg. The time course of binocular rivalry reveals a fundamental role of noise. *Journal of Vision*, 6(11):8, 2006.

[4] S.D. Brown and M. Steyvers. Detecting and predicting changes. *Cognitive Psychology*, 58(1):49–67, 2009.

[5] O.L. Carter, T.G. Campbell, G.B. Liu, and G. Wallis. Contradictory influence of context on predominance during binocular rivalry. *Clinical and Experimental Optometry*, 87:153–162, 2004.

[6] N.D. Daw and A.C. Courville. The pigeon as particle filter. *Advances in Neural Information Processing Systems*, 20, 2007.

[7] P. Dayan. A hierarchical model of binocular rivalry. *Neural Computation*, 10(5):1119–1135, 1998.

[8] S. Geman and D. Geman. Stochastic relaxation, Gibbs distributions, and the Bayesian restoration of images. *IEEE Transactions of Pattern Analysis and Machine Intelligence*, 6:721–741, 1984.

[9] T.L. Griffiths and J.B. Tenenbaum. Optimal predictions in everyday cognition. *Psychological Science*, 17(9):767–773, 2006.

[10] J. Hohwy, A. Roepstorff, and K. Friston. Predictive coding explains binocular rivalry: An epistemological review. *Cognition*, 108(3):687–701, 2008.

[11] T. Knapen, R. Kanai, J. Brascamp, J. van Boxtel, and R. van Ee. Distance in feature space determines exclusivity in visual rivalry. *Vision Research*, 47(26):3269–3275, 2007.

[12] S.H. Lee, R. Blake, and D.J. Heeger. Traveling waves of activity in primary visual cortex during binocular rivalry. *Nature Neuroscience*, 8(1):22–23, 2005.

[13] D.A. Leopold and N.K. Logothetis. Multistable phenomena: changing views in perception. *Trends in Cognitive Sciences*, 3(7):254–264, 1999.

[14] R.P. Levy, Reali. F., and T.L. Griffiths. Modeling the effects of memory on human online sentence processing with particle filters. *Advances in Neural Information Processing Systems*, 21:937, 2009.

[15] L. Liu, C.W. Tyler, and C.M. Schor. Failure of rivalry at low contrast: evidence of a suprathreshold binocular summation process. *Vision research*, 32(8):1471–1479, 1992.

[16] D.J.C. MacKay. *Information theory, inference and learning algorithms*. Cambridge University Press, 2003.

[17] P. Mamassian and R. Goutcher. Temporal dynamics in bistable perception. *Journal of Vision*, 5(4):7, 2005.

[18] N. Metropolis and S. Ulam. The Monte Carlo method. *Journal of the American Statistical Association*, pages 335–341, 1949.

[19] A.N. Sanborn, T.L. Griffiths, and D.J. Navarro. A more rational model of categorization. In *Proceedings of the 28th annual conference of the cognitive science society*, pages 726–731, 2006.

[20] P.R. Schrater and R. Sundareswara. Theory and dynamics of perceptual bistability. *Advances in Neural Information Processing Systems*, 19:1217, 2007.

[21] L. Shi, N.H. Feldman, and T.L. Griffiths. Performing bayesian inference with exemplar models. In *Proceedings of the 30th Annual Conference of the Cognitive Science Society*, pages 745–750, 2008.

[22] R. Sundareswara and P.R. Schrater. Perceptual multistability predicted by search model for Bayesian decisions. *Journal of Vision*, 8(5):12, 2008.

[23] E. Vul, N.D. Goodman, T.L. Griffiths, and J.B. Tenenbaum. One and done? Optimal decisions from very few samples. *Proceedings of the 31st Annual Meeting of the Cognitive Science Society*, 2009.

[24] E. Vul and H. Pashler. Measuring the crowd within: Probabilistic representations within individuals. *Psychological Science*, 19(7):645–647, 2008.

[25] H.R. Wilson. Minimal physiological conditions for binocular rivalry and rivalry memory. *Vision Research*, 47(21):2741–2750, 2007.

[26] H.R. Wilson, R. Blake, and S.H. Lee. Dynamics of travelling waves in visual perception. *Nature*, 412(6850):907–910, 2001.

